# Predicting Complex Behavior in Sparse Asymmetric Networks

**Ali A. Minai** and **William B. Levy**
Department of Neurosurgery
Box 420, Health Sciences Center
University of Virginia
Charlottesville, VA 22908

## Abstract

Recurrent networks of threshold elements have been studied intensively as associative memories and pattern-recognition devices. While most research has concentrated on fully-connected symmetric networks, which relax to stable fixed points, asymmetric networks show richer dynamical behavior, and can be used as sequence generators or flexible pattern-recognition devices. In this paper, we approach the problem of predicting the complex global behavior of a class of random asymmetric networks in terms of network parameters. These networks can show fixed-point, cyclical or effectively aperiodic behavior, depending on parameter values, and our approach can be used to set parameters, as necessary, to obtain a desired complexity of dynamics. The approach also provides qualitative insight into why the system behaves as it does and suggests possible applications.

## 1 INTRODUCTION

Recurrent neural networks of threshold elements have been intensively investigated in recent years, in part because of their interesting dynamics. Most of the interest has focused on networks with *symmetric connections*, which always relax to stable fixed points (Hopfield, 1982) and can be used as associative memories or pattern-recognition devices. Networks with *asymmetric connections*, however, have the potential for much richer dynamic behavior and may be used for learning sequences (see, e.g., Amari, 1972; Sompolinsky and Kanter, 1986).

In this paper, we introduce an approach for predicting the complex global behavior of an interesting class of random sparse asymmetric networks in terms of network parameters. This approach can be used to set parameter values, as necessary, to obtain a desired activity level and qualitatively different varieties of dynamic behavior.

## 2 NETWORK PARAMETERS AND EQUATIONS

A network consists of $n$ identical 0/1 neurons with threshold $\theta$. The fixed pattern of excitatory connectivity between neurons is generated prior to simulation by a Bernoulli process with a probability $p$ of connection from neuron $j$ to neuron $i$. All excitatory connections have the fixed value $w$, and there is a global inhibition that is linear in the number of active neurons. If $m(t)$ is the number of active neurons at time $t$, $K$ the inhibitory weight, $y_i(t)$ the net excitation and $z_i(t)$ the firing status of neuron $i$ at $t$, and $c_{ij}$ a 0/1 variable indicating the presence or absence of a connection from $j$ to $i$, then the equations for $i$ are:

$$y_i(t) = \frac{w \sum_{j=1}^{n} c_{ij} z_j(t-1)}{w \sum_{j=1}^{n} c_{ij} z_j(t-1) + Km(t-1)}, \quad 1 \le m(t-1) \le n \tag{1}$$

$$z_i(t) = \begin{cases} 1 & \text{if } y_i(t) \ge \theta \\ 0 & \text{otherwise} \end{cases}, \quad 0 < \theta < 1 \tag{2}$$

If $m(t-1)=0$, $y_i(t)=0 \ \forall i$. Equation (1) is a simple variant of the *shunting inhibition* neuron model studied by several researchers, and the network is similar to the one proposed by Marr (Marr, 1971). Note that (1) and (2) can be combined to write the neuron equations in a more familiar *subtractive inhibition* format. Defining $\alpha \equiv \theta K / (1-\theta)w$,

$$z_i(t) = \begin{cases} 1 & \text{if } \sum_{j=1}^{n} c_{ij} z_j(t-1) - \alpha \sum_{j=1}^{n} z_j(t-1) \ge 0 \\ 0 & \text{otherwise} \end{cases} \tag{3}$$

## 3 NETWORK BEHAVIOR

In this paper, we study the evolution of total activity, $m(t)$, as the system relaxes. From Equation (3), the firing condition for neuron $i$ at time $t$, given the activity $m(t-1)=M$ at time $t-1$, is: $e_i(t) \equiv \sum_{j=1}^{n} c_{ij} z_j(t-1) \ge \alpha M$. Thus, in order to fire at time $t$, neuron $i$ must have at least $\lceil \alpha M \rceil$ active inputs. This allows us to calculate the average firing probability of a neuron given the prior activity $M$ as:

$$P\{\# \text{ of active inputs} \ge \lceil \alpha M \rceil\} = \sum_{k=\lceil \alpha M \rceil}^{M} \binom{M}{k} p^k (1-p)^{M-k} \equiv \rho(M;n,p,\alpha) \tag{4}$$

If $M$ is large enough, we can use a Gaussian approximation to the binomial distribution

and a hyperbolic tangent approximation to the error function to get

$$\rho(M; n, p, \alpha) \approx \frac{1}{2}\left[1 - erf\left[\frac{X}{\sqrt{2}}\right]\right] \approx \frac{1}{2}\left[1 - \tanh\left[\sqrt{\frac{2}{\pi}}X\right]\right] \tag{5}$$

where

$$X \equiv \frac{\lceil \alpha M \rceil - Mp}{\sqrt{Mp(1-p)}}$$

Finally, when $M$ is large enough to assume $\lceil \alpha M \rceil \approx \alpha M$, we get an even simpler form:

$$\rho(M; n, p, \alpha) \approx \frac{1}{2}\left[1 - \tanh\frac{\sqrt{M}}{T}\right] \tag{6}$$

where

$$T \equiv \frac{1}{\alpha - p}\sqrt{\frac{\pi p(1-p)}{2}}, \qquad \alpha \neq p$$

Assuming that neurons fire independently, as they will tend to do in such large, sparse networks (Minai and Levy, 1992a,b), the network's activity at time $t$ is distributed as

$$P\{m(t)=N \mid m(t-1)=M\} \approx \binom{n}{N}\rho(M)^N(1-\rho(M))^{n-N} \tag{7}$$

which leads to a stochastic return map for the activity:

$$m(t) = n\rho(m(t-1)) + O(\sqrt{n}) \tag{8}$$

In Figure 1, we plot $m(t)$ against $m(t-1)$ for a 120 neuron network and two different values of $\alpha$. The vertical bars show two standard deviations on either side of $n\rho(m(t-1))$. It is clear that the network's activity falls within the range predicted by (8).

After an initial transient period, the system either switches off permanently (corresponding to the zero activity fixed point) or gets trapped in an $O(\sqrt{n})$ region around the point $\overline{m}$ defined by $m(t) = m(t-1)$. We call this the *attracting region* of the map. The size and location of the attracting region are determined by $\alpha$ and largely dictate the qualitative dynamic behavior of the network.

As $\alpha$ ranges from 0 to 1, networks show three kinds of behavior: fixed points, short cycles, and effectively aperiodic dynamics. Before describing these behaviors, however, we introduce the notion of *available neurons*. Let $k_i$ be the number of input connections to neuron $i$ (the *fan-in* of $i$). Given $m(t-1) = M$, if $k_i < \lceil \alpha M \rceil$, neuron $i$ cannot possibly meet the firing criterion at time $t$. Such a neuron is said to be disabled by activity $M$. The group of neurons not disabled are considered available neurons. At any specific activity $M$, there is a unique set, $N_a(M)$, of available neurons in a given network, and only neurons from this set can be active at the next time step. Clearly, $N_a(M_1) \subseteq N_a(M_2)$ if $M_1 \geq M_2$. The average size of the available set at a given activity $M$ is

$$n_a(M; n, p, \alpha) \equiv n\left[1 - P\{k_i < \lceil \alpha M \rceil\}\right] = n\sum_{k=\lceil \alpha M \rceil}^{n}\binom{n}{k}p^k(1-p)^{n-k} \tag{9}$$

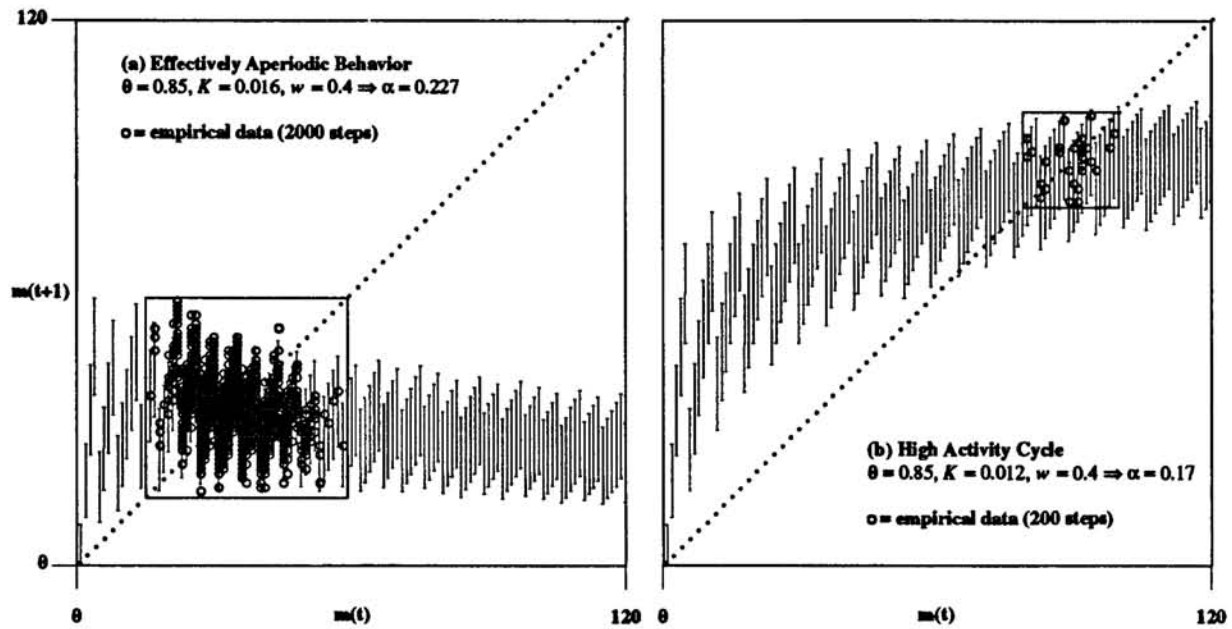

Figure 1: Predicted Distribution of $m(t+1)$ given $m(t)$, and Empirical Data (o) for Two Networks A and B. The vertical bars represent 4 standard deviations of the predicted distribution for each $m(t)$. Note that the empirical values fall in the predicted range.

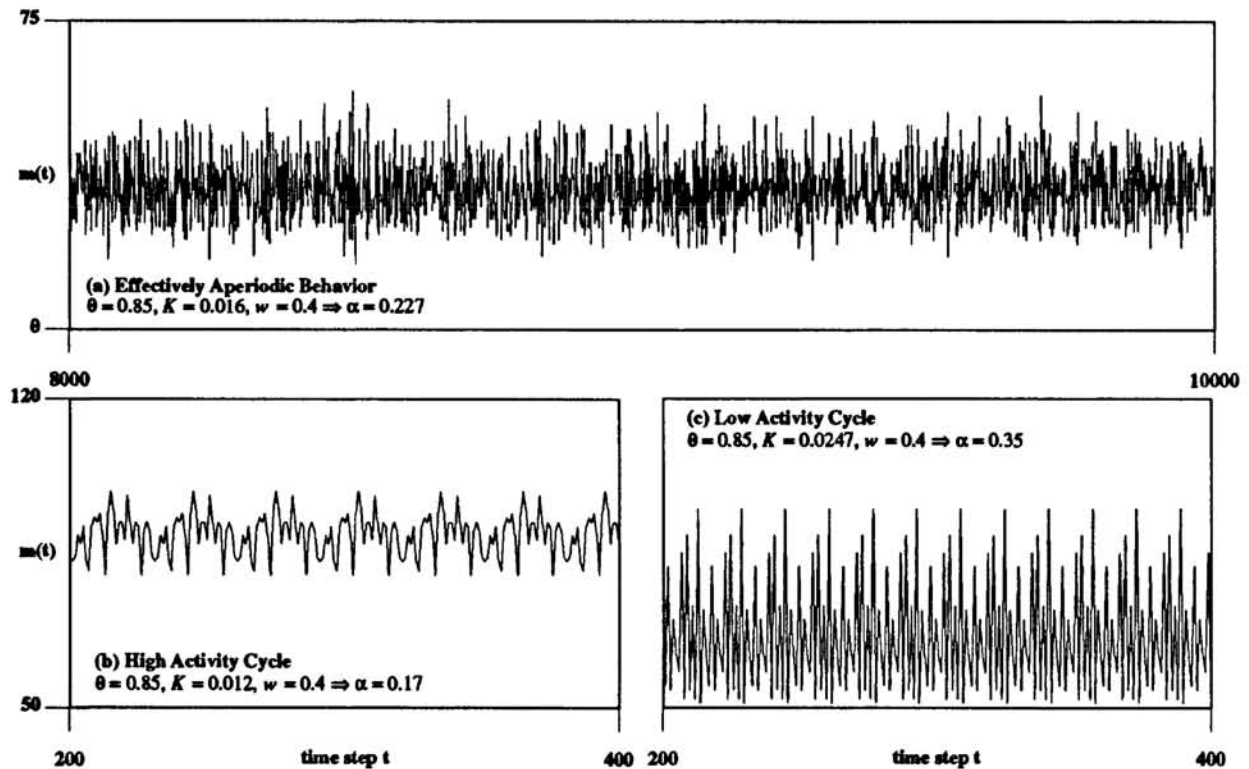

Figure 2: Activity time-series for three kinds of behavior shown by a 120 neuron network. Graphs (a) and (b) correspond to the data shown in Figure 1.

It can be shown that $n_a(M) \geq n\rho(M)$, so there are usually enough neurons available to achieve the average activity as per (8).

We now describe the three kinds of dynamic behavior exhibited by our networks.

(1) **Fixed Point Behavior:** If $\alpha$ is very small, $\bar{m}$ is close to $n$, inhibition is not strong enough to control activity and almost all neurons switch on permanently. If $\alpha$ is too large, $\bar{m}$ is close to 0 and the stochastic dynamics eventually finds, and remains at, the zero activity fixed point.

(2) **Effectively Aperiodic Behavior:** While deterministic, finite state systems such as our networks cannot show truly aperiodic or chaotic behavior, the time to repetition can be so long as to make the dynamics effectively aperiodic. This occurs when the attracting region is at a moderate activity level, well below the ceiling defined by the number of available neurons. In such a situation, the network, starting from an initial condition, successively visits a very large number of different states, and the activity, $m(t)$, yields an effvectively aperiodic time-series of amplitude $O(\sqrt{n})$, as shown in Figure 2(a).

(3) **Cyclical Behavior:** If the attracting region is at a high activity level, most of the available neurons must fire at every time step in order to maintain the activity predicted by (8). This forces network states to be very similar to each other, which, in turn, leads to even more similar successor states and the network settles into a relatively short limit cycle of high activity (Figure 2(b)). When the attracting region is at an activity level just above switch-off, the network can get into a low-activity limit cycle mediated by a very small group of high fan-in neurons (Figure 2(c)). This effect, however, is unstable with regard to initial conditions and the value of $\alpha$; it is expected to become less significant with increasing network size.

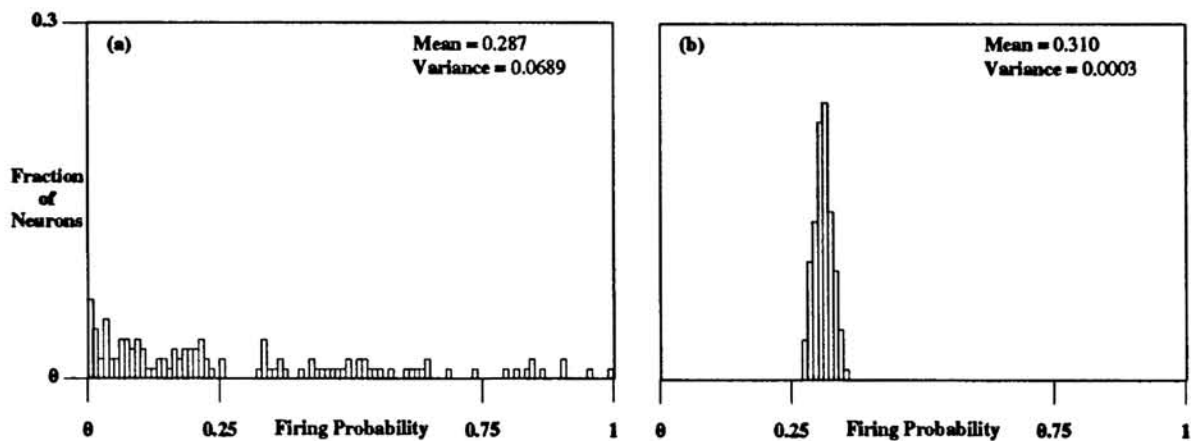

Figure 3: Neuron firing probability histograms for two 120-neuron networks in the effectively aperiodic phase ($\alpha \doteq 0.227$). Graph (a) is for a network with random connectivity generated through a Bernoulli process with $p = 0.2$, while Graph (b) is for a network with a fixed fan-in of exactly 24, which corresponds to the mean fan-in for $p = 0.2$.

One interesting issue that arises in the context of effectively aperiodic behavior is that of state-space sampling within the $O(\sqrt{n})$ constraint on activity. We assess this by looking at the histogram of individual neuron firing rates. Figure 3(a) shows the histogram for a 120 neuron network in the effectively aperiodic phase. Clearly, some subspaces are being sampled much more than others and the histogram is very broad. This is mainly due to differences in the fan-in of individual neurons, and will diminish in larger networks. Figure 3(b) shows the neuron firing histogram for a 120 neuron network where each neuron has a fan-in of 24. The sampling is clearly much more "ergodic" and the dynamics less biased towards certain subspaces.

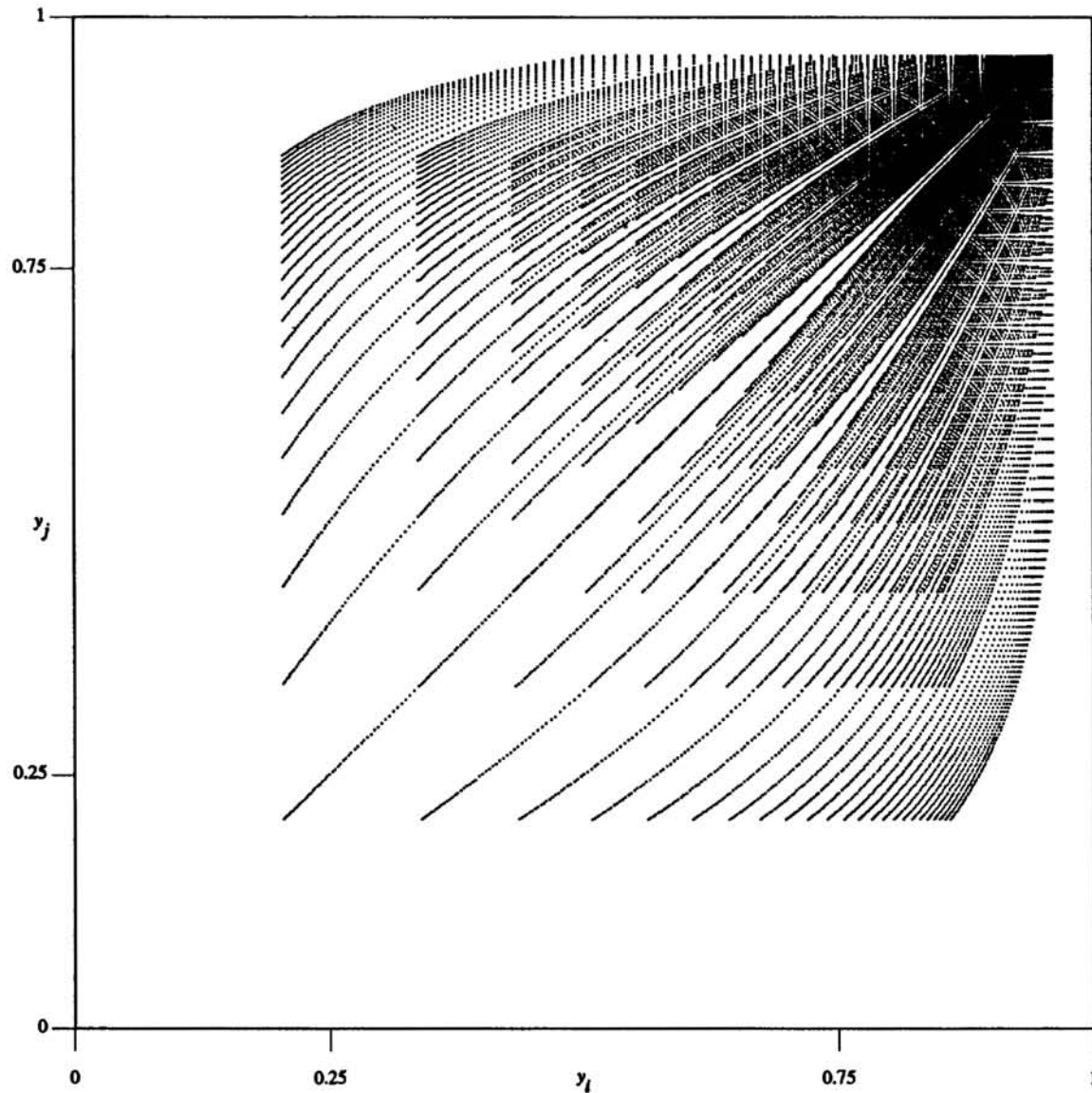

Figure 4: The complete set of non-zero activation values available to two identical neurons $i$ and $j$ with fan-in 24 in a 120-neuron network.

## 4 ACTIVATION DYNAMICS

While our modeling so far has focused on neural firing, it is instructive to look at the underlying neuron activation values, $y_i$. If $m(t-1) = M$, the possible $y_i(t)$ values for a neuron $i$ with fan-in $k_i$ are given by the set

$$Y(M, k_i) \equiv \left\{ \frac{wq}{wq + KM} \mid MAX(0, k_i - n + M) \leq q \leq MIN(M, k_i) \right\} \quad M > 0 \quad (10)$$

with $Y(0, k_i) \equiv \{0\}$. Here $q$ represents the number of active inputs to $i$, and the set $Y_i \equiv \bigcup_{M=0}^{n} Y(M, k_i)$ represents the set of all possible activation values for the neuron. The network's $n$-dimensional activation state, $\mathbf{y}(t) \equiv [y_1, y_2, ..., y_n]$, evolves upon the *activation space* $Y_1 \times Y_2 \times \cdots \times Y_n$, which is an extremely complex but regular object. In Figure 4, we plot a 2-dimensional subspace projection — called a $y$–$y$ plot – of the activation space for a 120-neuron network excluding the zero states. Both neurons shown have a fan-in of 24. In actuality, only a small subset of the activation space is sampled due to the constraining effects of the dynamics and the improbability of most $q$ values.

## 5 RELATING THE ACTIVITY LEVEL TO $\alpha$

From a practical standpoint, it would be useful to know how the average activity in a network is related to its $\alpha$ parameter. This can be done using the hyperbolic tangent approximation of Equation (6). First, we define the *activity level* at time $t$ as $r(t) \equiv n^{-1}m(t)$, i.e., the proportion of active neurons. This is a *macrostate* variable in the sense of (Amari, 1974). In the long term, the activity level becomes confined to a $O(1/\sqrt{n})$ region around the value corresponding to the activity fixed point. Thus, it is reasonable to use $\bar{r}$ as an estimate for the time-averaged activity level $\langle r \rangle$. To relate $\bar{m}$ (and thus $\bar{r}$) to $\alpha$, we must solve the fixed point equation $\bar{m} = n\rho(\bar{m})$. Substituting this and the definition of $\bar{r}$ into

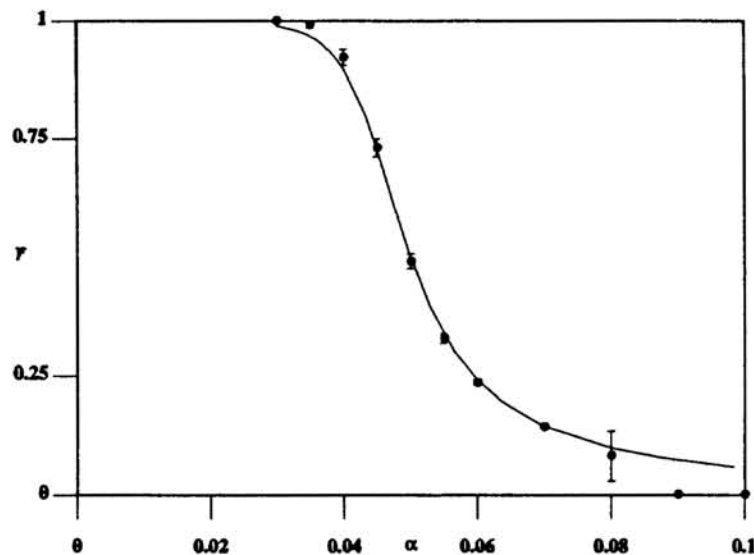

Figure 5: Predicted and empirical activities for 1000 neuron networks with $p = 0.05$. Each data point is averaged over 7 networks.

(6) gives:

$$\alpha(\bar{r}) \approx p + \sqrt{\frac{\pi p (1-p)}{2n\bar{r}}} \tanh^{-1}(1 - 2\bar{r})  \qquad (11)$$

While $\alpha$ can range from 0 to 1, the approximation of (11) breaks down at very high or very small values of $\bar{r}$. However, the range of its applicability gets wider as $n$ increases. Figure 5 shows the performance of (11) in predicting the average activity level in a 1000-neuron network. Note that $\alpha = p$ always leads to $\bar{r} = 0.5$ by Equation (11).

## 6 CONCLUSION

We have studied a general class of asymmetric networks and have developed a statistical model to relate its dynamical behavior to its parameters. This behavior, which is largely characterized by a composite parameter $\alpha$, is richly varied. Understanding such behavior provides insight into the complex possibilities offered by sparse asymmetric networks, especially with regard to modeling such brain regions as the hippocampal CA3 area in mammals. The complex behavior of random asymmetric networks has been discussed before by Parisi (Parisi, 1986), Nützel (Nützel, 1991), and others. We show how to *control* this complexity in our networks by setting parameters appropriately.

**Acknowledgements:** This research was supported by NIMH MH00622 and NIMH MH48161 to WBL, and by the Department of Neurosurgery, University of Virginia, Dr. John A. Jane, Chairman.

**References**

S. Amari (1972). Learning Patterns and Pattern Sequences by Self-Organizing Nets of Threshold Elements. *IEEE Trans. on Computers* C-21, 1197-1206

S. Amari (1974). A Method of Statistical Neurodynamics. *Kybernetik* 14, 201-215

J.J. Hopfield (1982). Neural Networks and Physical Systems with Emergent Collective Computational Abilities. *Proc. Nat. Acad. Sci. USA* 79, 2554-2558.

D. Marr (1971). Simple Memory: A Theory for Archicortex. *Phil. Trans. R. Soc. Lond. B* 262, 23-81.

A.A. Minai and W.B. Levy (1992a). The Dynamics of Sparse Random Networks. *In Review.*

A.A. Minai and W.B. Levy (1992b). Setting the Activity Level in Sparse Random Networks. *In Review.*

K. Nützel (1991). The Length of Attractors in Asymmetric Random Neural Networks with Deterministic Dynamics. *J. Phys. A: Math. Gen* 24, L151-157.

G. Parisi (1982). Asymmetric Neural Networks and the Process of Learning. *J. Phys. A: Math. Gen.* 19, L675-L680.

H. Sompolinsky and I. Kanter (1986), Temporal Association in Asymmetric Neural Networks. *Phys. Rev. Lett.* 57, 2861-2864.